# Methods Towards Invasive Human Brain Computer Interfaces

**Thomas Navin Lal**[1]**, Thilo Hinterberger**[2]**, Guido Widman**[3]**,
Michael Schröder**[4]**, Jeremy Hill**[1]**, Wolfgang Rosenstiel**[4]**,
Christian E. Elger**[3]**, Bernhard Schölkopf**[1] **and Niels Birbaumer**[2,5]

[1] Max-Planck-Institute for Biological Cybernetics, Tübingen, Germany
`{navin,jez,bs}@tuebingen.mpg.de`
[2] Eberhard Karls University, Dept. of Medical Psychology and
Behavioral Neurobiology, Tübingen, Germany
`{thilo.hinterberger,niels.birbaumer}@uni-tuebingen.de`
[3] University of Bonn, Department of Epileptology, Bonn, Germany
`{guido.widman,christian.elger}@ukb.uni-bonn.de`
[4] Eberhard Karls University, Dept. of Computer Engineering, Tübingen,
Germany `{schroedm,rosenstiel}@informatik.uni-tuebingen.de`
[5] Center for Cognitive Neuroscience, University of Trento, Italy

## Abstract

During the last ten years there has been growing interest in the development of Brain Computer Interfaces (BCIs). The field has mainly been driven by the needs of completely paralyzed patients to communicate. With a few exceptions, most human BCIs are based on extracranial electroencephalography (EEG). However, reported bit rates are still low. One reason for this is the low signal-to-noise ratio of the EEG [16]. We are currently investigating if BCIs based on electrocorticography (ECoG) are a viable alternative. In this paper we present the method and examples of intracranial EEG recordings of three epilepsy patients with electrode grids placed on the motor cortex. The patients were asked to repeatedly imagine movements of two kinds, e.g., tongue or finger movements. We analyze the classifiability of the data using Support Vector Machines (SVMs) [18, 21] and Recursive Channel Elimination (RCE) [11].

## 1 Introduction

Completely paralyzed patients cannot communicate despite intact cognitive functions. The disease Amyotrophic Lateral Sclerosis (ALS) for example, leads to complete paralysis of the voluntary muscular system caused by the degeneration of the motor neurons. Birbaumer et al. [1, 9] developed a Brain Computer Interface (BCI), called the Thought Translation Device (TTD), which is used by several paralyzed patients. In order to use the interface, patients have to learn to voluntary regulate their Slow Cortical Potentials (SCP). The system then allows its users to write text on the screen of a computer or to surf the web. Although it presents a major breakthrough, the system has two disadvantages. Not all patients manage

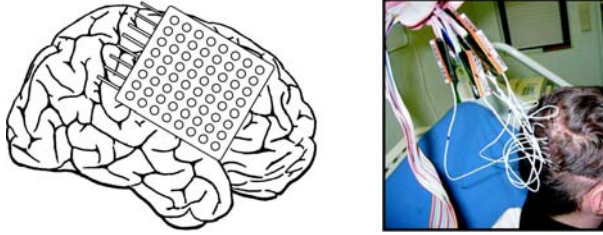

Figure 1: The left picture schematically shows the position of the 8x8 electrode grid of patient II. It was placed on the right hemisphere. As shown in the right picture the electrodes are connected to the amplifier via cables that are passed through the skull.

to control their SCP. Furthermore the bit rate is quite low. A well-trained user requires about 30 seconds to write one character.

Recently there has been increasing interest on EEG-based BCIs in the machine learning community. In contrast to the TTD, in many BCI-systems the computer learns rather than the system's user [2, 5, 11]. Most such BCIs require a data collection phase during which the subject repeatedly produces brain states of clearly separable locations. Machine learning techniques like Support Vector Machines or Fisher Discriminant are applied to the data to derive a classifying function. This function can be used in online applications to identify the different brain states produced by the subject.

The majority of BCIs is based on extracranial EEG-recordings during imagined limb movements. We restrict ourselves to mentioning just a few publications [14, 15, 17, 22]. Movement-related cortical potentials in humans on the basis of electrocorticographical data have also been studied, e.g. by [20]. Very recently the first work describing BCIs based on electrocorticographic recordings was published [6, 13]. Successful approaches have been developed using BCIs based on single unit, multiunit or field potentials recordings of primates. Serruya et al. taught monkeys to control a cursor on the basis of potentials from 7-30 motor cortex neurons [19]. The BCI developed by [3] enables monkeys to reach and grasp using a robot arm. Their system is based on recordings from frontoparietal cell ensembles.

Driven by the success of BCIs for primates based on single unit or multiunit recordings, we are currently developing a BCI-system that is based on ECoG recordings, as described in the present paper.

## 2 Electrocorticography and Epilepsy

All patients presented suffer from a focal epilepsy. The epileptic focus - the part of the brain which is responsible for the seizures - is removed by resection. Prior to surgery, the epileptic focus has to be localized. In some complicated cases, this must be done by placing electrodes onto the surface of the cortex as well as into deeper regions of the brain. The skull over the region of interest is removed, the electrodes are positioned and the incision is sutured. The electrodes are connected to a recording device via cables (cf. Figure 1). Over a period of a 5 to 14 days ECoG is continuously recorded until the patient has had enough seizures to precisely localize the focus [10]. Prior to surgery the parts of the cortex that are covered by the electrodes are identified by the electric stimulation of electrodes.

In the current setup, the patients keep the electrode implants for one to two weeks. After the implantation surgery, several days of recovery and follow-up examinations are needed. Due to the tight time constraints, it is therefore not possible to run long experiments. Furthermore most of the patients cannot concentrate for a long period of time. Therefore only a small amount of data could be collected.

Table 1: Positions of implanted electrodes. All three patients had an electrode grid implanted that partly covered the right or the left motor cortex.

| patient | implanted electrodes | task | trials |
|---|---|---|---|
| I | 64-grid right hemisphere, two 4-strip interhemisphere | left vs. right hand | 200 |
| II | 64-grid right hemisphere | little left finger vs. tongue | 150 |
| III | 20-grid central, four 16-strips frontal | little right finger vs. tongue | 100 |

## 3 Experimental Situation and Data Acquisition

The experiments were performed in the department of epileptology of the University of Bonn. We recorded ECoG data from three epileptic patients with a sampling rate of 1000Hz.

The electrode grids were placed on the cortex under the dura mater and covered the primary motor and premotor area as well as the fronto-temporal region either of the right or left hemisphere. The grid-sizes ranged from 20 to 64 electrodes. Furthermore two of the patients had additional electrodes implanted on other parts of the cortex (cf. Table 1). The imagery tasks were chosen such that the involved parts of the brain

- were covered by the electrode grid
- were represented spatially separate in the primary motor cortex.

The expected well-localized signal in motor-related tasks suggested discrimination tasks using imagination of hand, little finger, or tongue movements.

The patients were seated in a bed facing a monitor and were asked to repeatedly imagine two different movements. At the beginning of each trial, a small fixation cross was displayed in the center of the screen. The 4 second imagination phase started with a cue that was presented in the form of a picture showing either a tongue or a little finger for patients II and III. The cue for patient I was an arrow pointing left or right. There was a short break between the trials. The images which were used as a cue are shown in Figure 5.

## 4 Preprocessing

Starting half a second after the visualization of the task-cue, we extracted a window of length 1.5 seconds from the data of each electrode. For every trial and every electrode we thus obtained an EEG sequence that consisted of 1500 samples. The linear trend from every sequence was removed. Following [8, 11, 15] we fitted a forward-backward autoregressive model of order three to each sequence. The concatenated model parameters of the channels together with the descriptor of the imagined task (i.e. +1, -1) form one training point. For a given number $n$ of EEG channels, a training point $(x, y)$ is therefor a point in $\mathbb{R}^{3n} \times \{-1, 1\}$.

## 5 Channel Selection

The number of available training points is relatively small compared to the dimensionality of the data. The data of patient III for example, consists of only 100 training points of

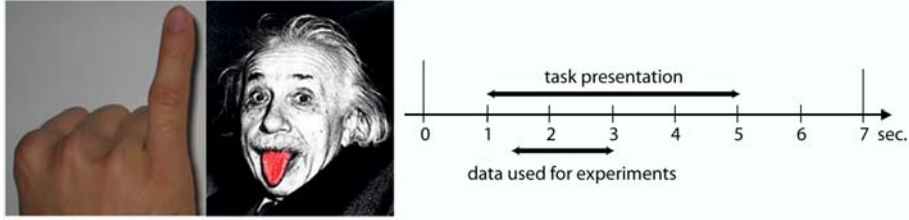

Figure 2: The patients were asked to repeatedly imagine two different movements that are represented separately at the primary cortex, e.g. tongue and little finger movements. This figure shows two stimuli that were used as a cue for imagery. The trial structure is shown on the right. The imagination phase lasted four seconds. We extracted segments of 1.5 seconds from the ECoG recordings for the analysis.

dimension 252. This is a typical setting in which features selection methods can improve classification accuracy.

Lal et al. [11] recently introduced a feature selection method for the special case of EEG data. Their method is based on Recursive Feature Elimination (RFE) [7]. RFE is a backward feature selection method. Starting with the full data set, features are iteratively removed from the data until a stopping criteria is met. In each iteration a Support Vector Machine (SVM) is trained and its weight vector is analyzed. The feature that corresponds to the smallest weight vector entry is removed.

Recursive Channel Elimination (RCE) [11] treats features that belong to the data of a channel in a consistent way. As in RFE, in every iteration one SVM is trained. The evaluation criteria that determines which of the remaining channels will be removed is the mean of the weight vector entries that correspond to a channel's features. All features of the channel with the smallest mean value are removed from the data. The output of RCE is a list of ranked channels.

## 6 Data Analysis

To begin with, we are interested in how well SVMs can *learn* from small ECoG data sets. Furthermore we would like to understand how localized the classification-relevant information is, i.e. how many recording positions are necessary to obtain high classification accuracy. We compare how well SVMs can generalize given the data of different subsets of ECoG-channels:

   (i) the complete data, i.e. all channels

  (ii) the subset of channels suggested by RCE. In this setting we use the list of ranked channels from RCE in the following way: For every $l$ in the range of one to the total number of channels, we calculate a 10-fold cross-validation error on the data of the $l$ best-ranked channels. We use the subset of channels which leads to the lowest error estimate.

 (iii) the two best-ranked channels by RCE. The underlying assumption used here is that the classification-relevant information is extremely localized and that two correctly chosen channels contain sufficient information for classification purposes.

 (iv) two channels drawn at random.

Throughout the paper we use linear SVMs. For regularization purposes we use a ridge on the kernel matrix which corresponds to a 2-norm penalty on the slack variables [4].

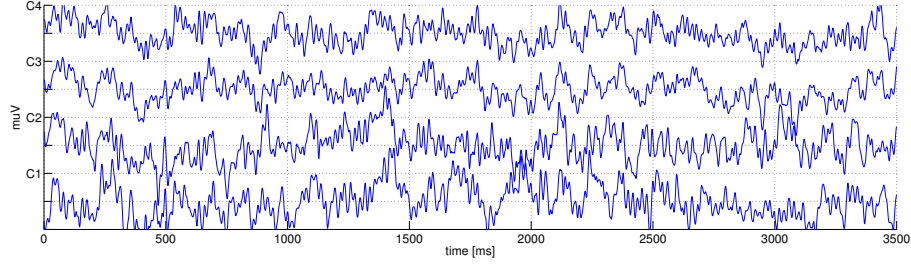

Figure 3: This plot shows ECoG recordings from 4 channels while the patient was imagining movements. The distance of two horizontal lines decodes $100\mu V$. The amplitude of the recordings ranges roughly from -100 $\mu V$ to +100 $\mu V$ which is on the order of five to ten times the amplitude measured with extracranial EEG.

To evaluate the classification performance of an SVM that is trained on a specific subset of channels we calculate its prediction error on a separate test set. We use a double-cross-validation scheme - the following procedure is repeated 50 times:
We randomly split the data into a training set (80%) and a test set (20%). Via 10-fold cross-validation on the training set we estimate all parameters for the different considered subsets (i)-(iv):

(i) The ridge is estimated.

(ii) On the basis of the training set RCE suggests a subset of channels. We restrict the training set as well as the test set to these channels. A ridge-value is then estimated from the restricted training set.

(iii) We restrict the training set and the test set to the 2 best ranked channels by RCE. The ridge is then estimated on the restricted training set.

(iv) The ridge is estimated.

We then train an SVM on the (restricted) training set using the estimated ridge. The trained model is tested on the (restricted) test set. For (i)-(iv) we obtain 50 test error estimates from the 50 repetitions for each patient. Table 2 summarizes the results.

## 7 Results

The results in Table 2 show that the generalization ability can significantly be increased by RCE. For patient I the error decreases from 38% to 24% when using the channel subsets suggested by RCE. In average RCE selects channel subsets of size 5.8. For patient II the number of channels is reduced to one third but the channel selection process does not yield an increased accuracy. The error of 40% can be reduced to 23% for patient III using in average 5 channels selected by RCE.
For patients I and III the choice of the best 2 ranked channels leads to a much lower error as well. The direct comparison of the results using the two best ranked channels to two randomly chosen channels shows how well the RCE ranking method works: For patient three the error drops from chance level for two random channels to 18 % using the two best-ranked channels.

The reason why there is such a big difference in performance for patient III when comparing (i) and (iii) might be, that out of the 84 electrodes, only 20 are located over or close to the motor cortex. RCE successfully identifies the important electrodes.
In contrast to patient III, the electrodes of patient II are all more or less located close to

Table 2: **Classification Results.** We compare the classification accuracy of SVMs trained on the data of different channel subsets: (i) all ECoG-channels, (ii) the subset determined by Recursive Channel Elimination (RCE), (iii) the subset consisting of the two best ranked channels by RCE and (iv) two randomly drawn channels. The mean errors of 50 repetitions are given along with the standard deviations. The test error can significantly be reduced by RCE for two of the three patients. Using the two best ranked channels by RCE also yields good results for two patients. SVMs trained on two random channels show performance better than chance only for patient II.

| pat | all channels (i) | | RCE cross-val. (ii) | | RCE top 2 (iii) | random 2 (iv) |
|-----|------------------|------|--------------------|------|-----------------|---------------|
|     | #channels | error | #channels | error | error | error |
| I   | 74 | $0.382 \pm 0.071$ | 5.8 | $0.243 \pm 0.063$ | $0.244 \pm 0.078$ | chance level |
| II  | 64 | $0.257 \pm 0.076$ | 21.5 | $0.268 \pm 0.080$ | $0.309 \pm 0.086$ | $0.419 \pm 0.123$ |
| III | 84 | $0.4 \pm 0.1$ | 5.0 | $0.233 \pm 0.13$ | $0.175 \pm 0.078$ | chance level |

the motor cortex. This explains why data from two randomly drawn channels can yield a classification rate better than chance. Furthermore patient II had the fewest electrodes implanted and thus the chance of randomly choosing an electrode close to an important location is higher than for the other two patients.

## 8 Discussion

We recorded ECoG-data from three epilepsy patients during a motor imagery experiment. Although only few data were collected, the following conclusions can be drawn:

- The data of all three patients is reasonably well classifiable. The error rates range from 17.5% to 23.3%. This is still high compared to the best error rates from BCI based on extracranial EEG which are as low as 10% (e.g. [12]). Please note that we used 1.5 seconds data from each trial only and that very few training points (100-200) were available. Furthermore, extracranial EEG has been studied and developed for a number of years.

- Recursive Channel Elimination (RCE) shows very good performance. RCE successfully identifies subsets of ECoG-channels that lead to good classification performance. On average, RCE leads to a significantly improved classification rate compared to a classifier that is based on the data of all available channels.

- Poor classification rates using two randomly drawn channels and high classification rates using the two best-ranked channels by RCE suggest that classification relevant information is focused on small parts of the cortex and depends on the location of the physiological function.

- The best ranked RCE-channels correspond well with the results from the electric stimulation (cf. Figure 8).

## 9 Ongoing Work and Further Research

Although our preliminary results indicate that invasive Brain Computer Interfaces may be feasible, a number of questions need to be investigated in further experiments. For instance, it is still an open question whether the patients are able to adjust to a trained classifier and whether the classifying function can be transferred from session to session. Moreover, experiments that are based on tasks different from motor imaginary need to

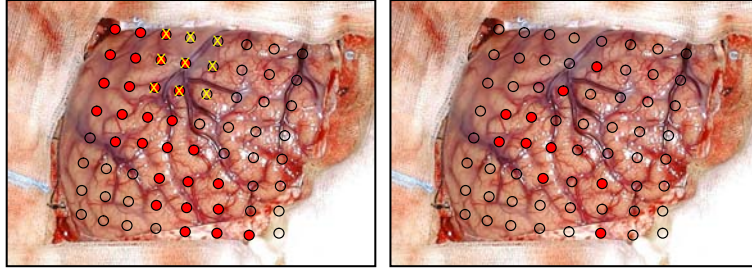

Figure 4: Electric stimulation of the implanted electrodes helps to identify the parts of the cortex that are covered by the electrode grid. This information is necessary for the surgery. The red (solid) dots on the left picture mark the motor cortex of patient II as identified by the electric stimulation method. The positions marked with yellow crosses correspond to the epileptic focus. The red points on the right image are the best ranked channels by Recursive Channel Elimination (RCE). The RCE-channels correspond well to the results from the electro stimulation diagnosis.

be implemented and tested. It is quite conceivable that the tasks that have been found to work well for extracranial EEG are not ideal for ECoG. Likewise, it is unclear whether our preprocessing and machine learning methods, originally developed for extracranial EEG data, are well adapted to the different type of data that ECoG delivers.

## Acknowledgements

This work was supported in part by the Deutsche Forschungsgemeinschaft (SFB 550, B5 and grant RO 1030/12), by the National Institute of Health (D.31.03765.2), and by the IST Programme of the European Community, under the PASCAL Network of Excellence, IST-2002-506778. T.N.L. was supported by a grant from the Studienstiftung des deutschen Volkes. Special thanks go to Theresa Cooke.

## References

[1] N. Birbaumer, N. Ghanayim, T. Hinterberger, I. Iversen, B. Kotchoubey, A. Kübler, J. Perelmouter, E. Taub, and H. Flor. A spelling device for the paralysed. *Nature*, 398:297–298, 1999.

[2] B. Blankertz, G. Curio, and K. Müller. Classifying single trial EEG: Towards brain computer interfacing. In T.K. Leen, T.G. Dietterich, and V. Tresp, editors, *Advances in Neural Information Processing Systems*, volume 14, Cambridge, MA, USA, 2001. MIT Press.

[3] J.M. Carmena, M.A Lebedev, R.E Crist, J.E O'Doherty, D.M. Santucci, D. Dimitrov, P.G. Patil, C.S Henriquez, and M.A. Nicolelis. Learning to control a brain-machine interface for reaching and grasping by primates. *PLoS Biology*, 1(2), 2003.

[4] C. Cortes and V. Vapnik. Support-vector networks. *Machine Learning*, 20:273–297, 1995.

[5] J. del R. Millan, F. Renkens, J. Mourino, and W. Gerstner. Noninvasive brain-actuated control of a mobile robot by human eeg. *IEEE Transactions on Biomedical Engineering. Special Issue on Brain-Computer Interfaces*, 51(6):1026–1033, June 2004.

[6] B. Graimann, J. E. Huggins, S. P. Levine, and G. Pfurtscheller. Towards a direct brain interface based on human subdural recordings and wavelet packet analysis. *IEEE Trans. IEEE Transactions on Biomedical Engineering*, 51(6):954–962, 2004.

[7] I. Guyon, J. Weston, S. Barnhill, and V. Vapnik. Gene selection for cancer classification using support vector machines. *Journal of Machine Learning Research*, 3:1439–1461, March 2003.

[8] S. Haykin. *Adaptive Filter Theory*. Prentice-Hall International, Inc., Upper Saddle River, NJ, USA, 1996.

[9] T. Hinterberger, J. Kaiser, A. Kbler, N. Neumann, and N. Birbaumer. The Thought Translation Device and its Applications to the Completely Paralyzed. In Diebner, Druckrey, and Weibel, editors, *Sciences of the Interfaces*. Genista-Verlag, Tübingen, 2001.

[10] J. Engel Jr. Presurgical evaluation protocols. In *Surgical Treatment of the Epilepsies*, pages 740–742. Raven Press Ltd., New York, 2nd edition, 1993.

[11] T.N. Lal, M. Schröder, T. Hinterberger, J. Weston, M. Bogdan, N. Birbaumer, and B. Schölkopf. Support Vector Channel Selection in BCI. *IEEE Transactions on Biomedical Engineering. Special Issue on Brain-Computer Interfaces*, 51(6):1003–1010, June 2004.

[12] S. Lemm, C. Schäfer, and G. Curio. BCI Competition 2003 - Data Set III: Probabilistic Modeling of Sensorimotor mu-Rhythms for Classification of Imaginary Hand Movements. *IEEE Transactions on Biomedical Engineering. Special Issue on Brain-Computer Interfaces*, 51(6):1077–1080, June 2004.

[13] E. C. Leuthardt, G. Schalk, J. R. Wolpaw, J. G. Ojemann, and D. W. Moran. A braincomputer interface using electrocorticographic signals in humans. *Journal of Neural Engineering*, 1:63–71, 2004.

[14] D.J. McFarland, L.M. McCane, S.V. David, and J.R. Wolpaw. Spatial filter selection for EEG-based communication. *Electroencephalography and Clinical Neurophysiology*, 103:386–394, 1997.

[15] G. Pfurtscheller., C. Neuper amd A. Schlögl, and K. Lugger. Separability of EEG signals recorded during right and left motor imagery using adaptive autoregressive parameters. *IEEE Transactions on Rehabilitation Engineering*, 6(3):316–325, 1998.

[16] J. Raethjen, M. Lindemann, M. Dümpelmann, R. Wenzelburger, H. Stolze, G. Pfister, C. E. Elger, J. Timmer, and G. Deuschl. Corticomuscular coherence in the 6-15 hz band: is the cortex involved in the generation of physiologic tremor? *Experimental Brain Research*, 142:32–40, 2002.

[17] H. Ramoser, J. Müller-Gerking, and G. Pfurtscheller. Optimal spatial filtering of single trial EEG during imagined hand movement. *IEEE Transactions on Rehabilitation Engineering*, 8(4):441–446, 2000.

[18] B. Schölkopf and A. Smola. *Learning with Kernels*. MIT Press, Cambridge, USA, 2002.

[19] M.D. Serruya, N.G Hatsopoulos, L. Paninski, M.R. Fellows, and Donoghue J.P. Instant neural control of a movement signal. *Nature*, 416:141–142, 2002.

[20] C. Toro, G. Deuschl, R. Thatcher, S. Sato, C. Kufta, and M. Hallett. Event-related desynchronization and movement-related cortical potentials on the ECoG and EEG. *Electroencephalography Clinical Neurophysiology*, 5:380–389, 1994.

[21] V. N. Vapnik. *Statistical Learning Theory*. John Wiley and Sons, New York, USA, 1998.

[22] R. Wolpaw and D.J McFarland. Multichannel EEG-based brain-computer communication. *Electroencephalography and Clinical Neurophysiology*, 90:444–449, 1994.
